# An MCMC-Based Method of Comparing Connectionist Models in Cognitive Science

**Woojae Kim, Daniel J. Navarro,** *Mark A. Pitt, In Jae Myung**
Department of Psychology
Ohio State University
{kim.1124, navarro.20, pitt.2, myung.1}@osu.edu

## Abstract

Despite the popularity of connectionist models in cognitive science, their performance can often be difficult to evaluate. Inspired by the geometric approach to statistical model selection, we introduce a conceptually similar method to examine the global behavior of a connectionist model, by counting the number and types of response patterns it can simulate. The Markov Chain Monte Carlo-based algorithm that we constructed finds these patterns efficiently. We demonstrate the approach using two localist network models of speech perception.

## 1   Introduction

Connectionist models are popular in some areas of cognitive science, especially language processing. One reason for this is that they provide a means of expressing the fundamental principles of a theory in a readily testable computational form. For example, levels of mental representation can be mapped onto layers of nodes in a connectionist network. Information flow between levels is then defined by the types of connection (e.g., excitatory and inhibitory) between layers. The soundness of the theoretical assumptions are then evaluated by studying the behavior of the network in simulations and testing its predictions experimentally.

Although this sort of modeling has enriched our understanding of human cognition, the consequences of the choices made in the design of a model can be difficult to evaluate. While good simulation performance is assumed to support the model and its underlying principles, a drawback of this testing methodology is that it can obscure the role played by a model's complexity and other reasons why a competing model might simulate human data equally well.

These concerns are part and parcel of the well-known problem of model selection. A great deal of progress has been made in solving it for statistical models (i.e., those that can be described by a family of probability distributions [1, 2]). Connectionist

models, however, are a computationally different beast. The current paper introduces a technique that can be used to assist in evaluating and choosing between connectionist models of cognition.

## 2 A Complexity Measure for Connectionist Models

The ability of a connectionist model to simulate human performance well does not provide conclusive evidence that the network architecture is a good approximation to the human cognitive system that generated the data. For instance, it would be unimpressive if it turned out that the model could also simulate many non-human-like patterns. Accordingly, we need a "global" view of the model's behavior to discover all of the qualitatively different patterns it can simulate.

A model's ability to reproduce diverse patterns of data is known as its complexity, an intrinsic property of a model that arises from the interaction between its parameters and functional form. For statistical models, it can be calculated by integrating the determinant of the Fisher information matrix over the parameter space of the model, and adding a term that is linear in the number of parameters. Although originally derived by Rissanen [1] from an algorithmic coding perspective, this measure is sometimes called the geometric complexity, because it is equal to the logarithm of the ratio of two Riemannian volumes. Viewed from this geometric perspective, the measure has an elegant interpretation as a count of the number of "distinguishable" distributions that a model can generate [3, 4]. Unfortunately, geometric complexity cannot be applied to connectionist models, because these models rarely possess a likelihood function, much less a well-defined Fisher information matrix. Also, in many cases a learning (i.e., model-fitting) algorithm for finding optimal parameter values is not proposed along with the model, further complicating matters.

A conceptually simple solution to the problem, albeit a computationally demanding one, is first to discretize the data space in some properly defined sense and then to identify all of the data patterns a connectionist model can generate. This approach provides the desired global view of the model's capabilities and its definition resembles that of geometric complexity: the complexity of a connectionist model is defined in terms of the number of discrete data patterns the model can produce. As such, this reparametrization-invariant complexity measure can be used for virtually all types of network models provided that the discretization of the data space is both justifiable and meaningful.

A challenge in implementing this solution lies in the enormity of the data space, which may contain a truly astronomical number of patterns. Only a small fraction of these might correspond to a model's predictions, so it is essential to use an efficient search algorithm, one that will find most or all of these patterns in a reasonable time. We describe an algorithm that uses Markov Chain Monte Carlo (MCMC) to solve such problems. It is tailored to exploit the kinds of search spaces that we suspect are typical of localist connectionist models, and we evaluate its performance on two of them.

## 3 Localist Models of Phoneme Perception

A central issue in the field of human speech perception is how lexical knowledge influences the perception of speech sounds. That is, how does knowing the word you are hearing influence how you hear the smaller units that make up the word (i.e., its phonemes)? Two localist models have been proposed that represent opposing theoretical positions. Both models were motivated by different theoretical prin-

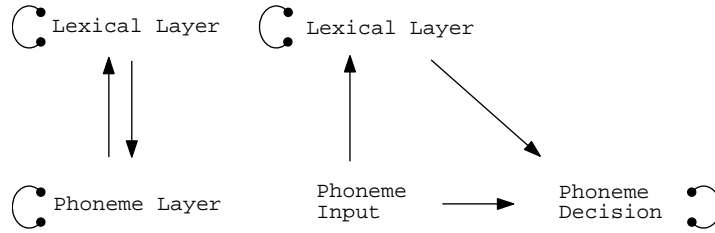

Figure 1: Network architectures for TRACE (left) and MERGE (right). Arrows indicate excitatory connections between layers; lines with dots indicate inhibitory connections within layers.

ciples. Proponents of TRACE [5] argue for bi-directional communication between layers whereas proponents of MERGE [6] argue against it. The models are shown schematically in Figure 1. Each contains two main layers. Phonemes are represented in the first layer and words in the second. Activation flows from the first to the second layer in both models. At the heart of the controversy is whether activation also flows in the reverse direction, directly affecting how the phonemic input is processed. In TRACE it can. In MERGE it cannot. Instead, the processing performed at the phoneme level in MERGE is split in two, with an input stage and a phoneme decision stage. The second, lexical layer cannot directly affect phoneme activation. Instead, the two sources of information (phonemic and lexical) are integrated only at the phoneme decision stage.

Although the precise details of the models are unnecessary for the purposes of this paper , it will be useful to sketch a few of their technical details. The parameters for the models (denoted $\theta$), of which TRACE has 7 and MERGE has 11, correspond to the strength of the excitatory and inhibitory connections between nodes, both within and between layers. The networks receive a continuous input, and stabilize at a final state after a certain number of cycles. In our formulation, a parameter set $\theta$ was considered valid only if the final state satisfied certain decision criteria (discussed shortly). Detailed descriptions of the models, including typical parameter values, are given by [5] and [6].

Despite the differences in motivation, TRACE and MERGE are comparable in their ability to simulate key experimental findings [6], making it quite challenging if not impossible to distinguish between then experimentally. Yet surely the models are not identical? Is one more complex than the other? What are the functional differences between the two?

In order to address these questions, we consider data from experiments by [6] which are captured well by both models. In the experiments, monosyllabic words were presented in which the last phoneme from one word was partially replaced by one from another word (through digital editing) to create word blends that retained residual information about the identity of the phoneme from both words. The six types of blends are listed on the left of Table 1. Listeners had to categorize the last phoneme in one task (phoneme decision) and categorize the entire utterance as a word or a nonsense word in the other task (lexical decision). The response choices in each task are listed in the table. Three responses choices were used in lexical decision to test the models' ability to distinguish between words, not just words and nonwords. The asterisks in each cell indicate the responses that listeners chose most often. Both TRACE and MERGE can simulate this pattern of responses.

Table 1: The experimental design. Asterisks denote human responses.

| Condition Name | Example | Phonemic Decision | | | | Lexical Decision | | |
|---|---|---|---|---|---|---|---|---|
| | | /b/ | /g/ | /z/ | /v/ | job | jog | nonword |
| bB | JOb + joB | * | | | | * | | |
| gB | JOg + joB | * | | | | * | | |
| vB | JOv + joB | * | | | | * | | |
| zZ | JOz + joZ | | | * | | | | * |
| gZ | JOg + joZ | | | * | | | | * |
| vZ | JOv + joZ | | | * | | | | * |

Table 2: Two sets of decision rules for TRACE and MERGE. The values shown correspond to activation levels of the appropriate decision node.

| Constraint | Phoneme Decision Choose /b/ if... | Lexical Decision Choose "job" if... | Choose "nonword" if... |
|---|---|---|---|
| Weak | /b/> 0.4 & others < 0.4 | job > 0.4 & jog < 0.4 | both < 0.4 |
| Strong | /b/> 0.45 & others < 0.25 (/b/ − max(others)) > 0.3 | job > 0.45 & jog < 0.25 (job − jog) > 0.3 | both < 0.25 abs(difference) < 0.15 |

The profile of responses decisions (phoneme and lexical) over the six experimental conditions provides a natural definition of a data pattern that the model could produce, and the decision rules establish a natural (surjective) mapping from the continuous space of network states (of which each model can produce some subset) to the discrete space of data patterns. We applied two different sets of decision rules, listed in Table 2, and were interested in determining how many patterns (besides the human-like pattern) each model can generate. As previously discussed, these counts will serve as a measure of model complexity.

## 4   The Search Algorithm

The search problem that we need to solve differs from the standard Monte Carlo counting problem. Ordinarily, Monte Carlo methods are used to discover how much of the search space is covered by some region by counting how often co-ordinates are sampled from that region. In our problem, a high-dimensional parameter space has been partitioned into an unknown number of regions, with each region corresponding to a single data pattern. The task is to find all such regions irrespective of their size. How do we solve this problem? Given the dimensionality of the space, brute force searches are impossible. Simple Monte Carlo (SMC; i.e., uniform random sampling) will fail because it ignores the structure of the search space.

The spaces that we consider seem to possess three regularities, which we call a "grainy" structure, illustrated schematically in Figure 2. Firstly, on many occasions the network does not converge on a state that meets the decision criteria, so some proportion of the parameter space does not correspond to any data pattern. Secondly, the size of the regions vary a great deal. Some data patterns are elicited by a wide range of parameter values, whereas others can be produced only by a small range of values. Thirdly, small regions tend to cluster together. In these models, there are likely to be regions where the model consistently chooses the dominant phoneme and makes the correspondingly appropriate lexical decision. However, there will also be large regions in which the models always choose "nonword" ir-

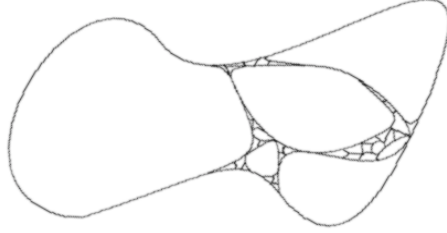

Figure 2: A parameter space with "grainy" structure. Each region corresponds to a single data pattern that the model can generate. Regions vary in size, and small regions cluster together.

respective of whether the stimulus is a word. Along the borders between regions, however, there might be lots of smaller "transition regions", and these regions will tend to be near one another.

The consequence of this structure is that the size of the region in which the process is currently located provides extensive information about the number of regions that are likely to lie nearby. In a small region, there will probably be other small regions nearby, so a fine-grained search is required in order to find them. However, a fine-grained search process will get stuck in a large region, taking tiny steps when great leaps are required. Our algorithm exploits this structure by using MCMC to estimate a different parameter sampling distribution $p(\theta|r_i)$ for every region $r_i$ that it encounters, and then cycling through these distributions in order to sample parameter sets. The procedure can be reduced to three steps:

1. Set $i = 0$, $m = 0$. Sample $\theta$ from $p(\theta|r_0)$, a uniform distribution over the space. If $\theta$ does not generate a valid data pattern, repeat Step 1.

2. Set $m = m + 1$ and then $i = m$. Record the new pattern, and use MCMC to estimate $p(\theta|r_i)$.

3. Sample $\theta$ from $p(\theta|r_i)$. If $\theta$ generates a new pattern, return to Step 2. Otherwise, set $i = \mathrm{mod}(i, m) + 1$, and repeat Step 3.

The process of estimating $p(\theta|r_i)$ is a fairly straightforward application of MCMC [7]. We specify a uniform jumping distribution over a small hypersphere centered on the current point $\theta$ in the parameter space, accepting candidate points if and only if they produce the same pattern as $\theta$. After collecting enough samples, we calculate the mean and variance-covariance matrix for these observations, and use this to estimate an ellipsoid around the mean, as an approximation to the $i$-th region. However, since we want to find points in the bordering regions, the the estimated ellipsoid is deliberately oversized. The sampling distribution $p(\theta|r_i)$ is simply a uniform distribution over the ellipsoid.

Unlike SMC (or even a more standard application of MCMC), our algorithm has the desirable property that it focuses on each region in equal proportion, irrespective of its size. Not only that, because the parameter space is high dimensional, the vast majority of the distribution $p(\theta|r_i)$ will actually lie near the edges of the ellipsoid: that is, the area just outside of the $i$-th region. Consequently, we search primarily along the edges of the regions that we have already discovered, paying closer attention to the small regions. The overall distribution $p(\theta)$ is essentially a mixture distribution that assigns higher density to points known to lie near many regions.

# 5 Testing the Algorithm

In the absence of analytic results, the algorithm was evaluated against standard SMC. The first test applied both to a simple toy problem possessing a grainy structure. Inside a hypercube $[0, 1]^d$, an assortment of large and small regions (also hypercubes) were defined using unevenly spaced grids so that all the regions neighbored each other ($d$ ranged from 3 to 6). In higher dimensions ($d \geq 4$), SMC did not find all of the regions. In contrast, the MCMC algorithm found all of the regions, and did so in a reasonable amount of time. Overall, the MCMC-based algorithm is slower than SMC at the beginning of the search due to the time required for region estimation. However, the time required to learn the structure of the parameter space is time well spent because the search becomes more efficient and successful, paying large dividends in time and accuracy in the end.

As a second test, we applied the algorithms to simplified versions of TRACE, constructed so that even SMC might work reasonably well. In one reduced model, for instance, only phoneme responses were considered. In the other, only lexical responses were considered. Weak and strong constraints (Table 2) were imposed on both models. In all cases, MCMC found as many or more patterns than SMC, and all SMC patterns were among the MCMC patterns.

# 6 Application to Models of Phoneme Perception

Next we ran the search algorithm on the full versions of TRACE and MERGE, using both the strong and weak constraints (Table 2). The number of patterns discovered in each case is summarized in Figure 3. In this experimental design MERGE is more complex than TRACE, although the extent of this effect is somewhat dependent on the choice of constraints. When strong constraints are applied TRACE (27 patterns) is nested within MERGE (67 patterns), which produces 148% more patterns. However, when these constraints are eased, the nesting relationship disappears, and MERGE (73 patterns) produces only 40% more patterns than TRACE (52 patterns). Nevertheless, it is noteworthy that the behavior of each is highly constrained, producing less than 100 of the $4^6 \times 3^6 = 2,985,984$ patterns available. Also, for both models (under both sets of constraints), the vast majority of the parameter space was occupied by only a few patterns.

A second question of interest is whether each model's ouput veers far from human performance (Table 1). To answer this, we classified every data pattern in terms of the number of mismatches from the human-like pattern (from 0 to 12), and counted how frequently the model patterns fell into each class. The results, shown in Figure 4, are quite similar and orderly for both models. The choice of constraints had little effect, and in both cases the TRACE distribution (open circles) is a little closer to the human-like pattern than the MERGE distribution (closed circles). Even so, both models are remarkably human-like when considered in light of the distribution of all possible patterns (cross hairs). In fact, the probability is virtually zero that a "random model" (consisting of a random sample of patterns) would display such a low mismatch frequency.

Building on this analysis, we looked for qualitative differences in the types of mismatches made by each model. Since the choice of constraints made no difference, Figure 5 shows the mismatch profiles under weak constraints. Both models produce no mismatches in some conditions (e.g., bB-phoneme identification, vZ-lexical decision) and many in others (e.g., gB-lexical decision). Interestingly, TRACE and MERGE produce similar mismatch profiles for lexical decision, and a comparable number of mismatches (108 vs. 124). However, striking qualitative differences are evident for

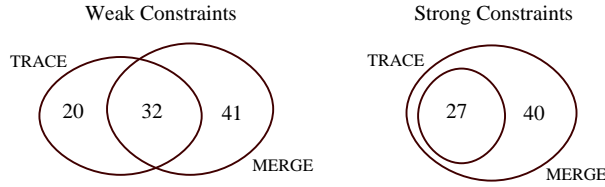

Figure 3: Venn diagrams showing the number of patterns discovered for both models under both types of constraint.

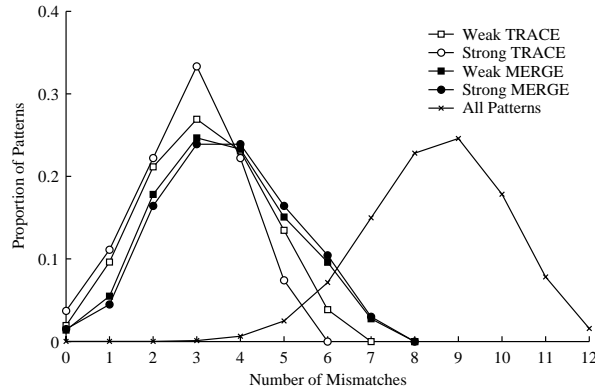

Figure 4: Mismatch distributions for all four models plus the data space. The 0-point corresponds to the lone human-like pattern contained in all distributions.

phoneme decisions, with MERGE producing mismatches in conditions that TRACE does not (e.g., vB, vZ). When the two graphs are compared, an asymmetry is evident in the frequency of mismatches across tasks: MERGE makes phonemic mismatches with about the same frequency as lexical errors (139 vs. 124), whereas TRACE does so less than half as often (56 vs. 108).

The mismatch asymmetry accords nicely with the architectures shown in Figure 1. The two models make lexical decisions in an almost identical manner: phonemic information feeds into the lexical decision layer, from which a decision is made. It should then come as no surprise that lexical processing in TRACE and MERGE is so similar. In contrast, phoneme processing is split between two layers in MERGE but confined to one in TRACE. The two layers dedicated to phoneme processing provide MERGE an added degree of flexibility (i.e., complexity) in generating data patterns. This shows up in many ways, not just in MERGE's ability to produce mismatches in more conditions than TRACE. For example, these mismatches yield a wider range of phoneme responses. Shown above each bar in Figure 5 is the phoneme that was misrecognized in the given condition. TRACE only misrecogized the phoneme as /g/ whereas MERGE misrecognized it as /g/, /z/, and /v/.

These analyses describe a few consequences of dividing processing between two layers, as in MERGE, and in doing so creating a more complex model. On the basis of performance (i.e., fit) alone, this additional complexity is unnecessary for modeling phoneme perception because the simpler architecture of TRACE simulates human data as well as MERGE. If MERGE's design is to be preferred, the additional complexity must be justifed for other reasons [6].

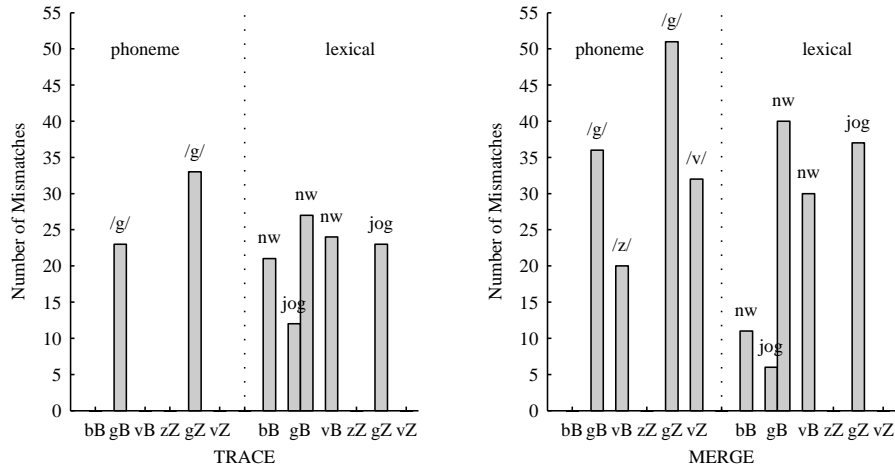

Figure 5: Mismatch profiles for both TRACE and MERGE when the weak constraints are applied. Conditions are denoted by their phoneme blend.

# 7    Conclusions

The results of this preliminary evaluation suggest that the MCMC-based algorithm is a promising method for comparing connectionist models. Although it was developed to compare localist models like TRACE and MERGE, it may be broadly applicable whenever the search space exhibits this "grainy" structure. Indeed, the algorithm could be a general tool for designing, comparing, and evaluating connectionist models of human cognition. Plans are underway to extend the approach to other experimental designs, dependent measures (e.g., reaction time), and models.

### Acknowledgements

The authors were supported by NIH grant R01-MH57472 awarded to IJM and MAP. DJN was also supported by a grant from the Office of Research at OSU. We thank Nancy Briggs, Cheongtag Kim and Yong Su for helpful discussions.

## Footnotes

*Correspondence should be addressed to Daniel Navarro, Department of Psychology, Ohio State University, 1827 Neil Avenue Mall, Columbus OH 43210, USA. Telephone: (614) 292-1030, Facsimile: (614) 292-5601.

### References

[1] Rissanen, J. (1996). Fisher information and stochastic complexity. *IEEE Transactions on Information Theory 42*, 40-47.

[2] Rissanen, J. (2001). Strong optimality of the normalized ML models as universal codes and information in data. *IEEE Transactions on Information Theory 47*, 1712-1717.

[3] Balasubramanian, V. (1997). Statistical inference, Occam's razor and statistical mechanics on the space of probability distributions. *Neural Computation, 9*, 349-368.

[4] Myung, I. J., Balasubramanian, V., & Pitt, M. A. (2000). Counting probability distributions: Differential geometry and model selection. *Proceedings of the National Academy of Sciences USA, 97*, 11170-11175.

[5] McClelland, J. L. & Elman, J. L. (1986). The TRACE model of speech perception. *Cognitive Psychology, 18*, 1-86.

[6] Norris, D., McQueen, J. M. & Cutler, A. (2000). Merging phonetic and lexical information in phonetic decision-making. *Behavioral & Brain Sciences, 23*, 299-325.

[7] Gilks, W. R. , Richardson, S., & Spiegelhalter, D. J. (1995). *Markov Chain Monte Carlo in Practice.* London: Chapman and Hall.